# Multiple Instance Learning on Structured Data

[1]**Dan Zhang,** [2]**Yan Liu,** [1]**Luo Si,** [3]**Jian Zhang,** [4]**Richard D. Lawrence**
1. Computer Science Department, Purdue University, West Lafayette, IN 47906
2. Computer Science Department, University of Southern California, Los Angeles, CA 90089
3. Statistics Department, Purdue University, West Lafayette, IN 47906
4. Machine Learning Group, IBM T.J. Watson Research Center, Yorktown Heights, NY 10598
[1]{zhang168, lsi}@cs.purdue.edu, [2]yanliu.cs@usc.edu, [3]jian.zhang@gmail.com , [4]ricklawr@us.ibm.com

## Abstract

Most existing *Multiple-Instance Learning (MIL)* algorithms assume data instances and/or data bags are independently and identically distributed. But there often exists rich additional dependency/structure information between instances/bags within many applications of MIL. Ignoring this structure information limits the performance of existing MIL algorithms. This paper explores the research problem as *multiple instance learning on structured data* (MILSD) and formulates a novel framework that considers additional structure information. In particular, an effective and efficient optimization algorithm has been proposed to solve the original non-convex optimization problem by using a combination of Concave-Convex Constraint Programming (CCCP) method and an adapted Cutting Plane method, which deals with two sets of constraints caused by learning on instances within individual bags and learning on structured data. Our method has the nice convergence property, with specified precision on each set of constraints. Experimental results on three different applications, i.e., webpage classification, market targeting, and protein fold identification, clearly demonstrate the advantages of the proposed method over state-of-the-art methods.

## 1 Introduction

*Multiple Instance Learning* (*MIL*) is a variation of the classical learning methods for problems with incomplete knowledge on the instances (or *examples*) [4]. In a MIL problem, the labels are assigned to bags, *i.e.,* a set of instances, rather than individual instances [1, 4, 5, 13]. MIL has been widely employed in areas such as text mining [1], drug design [4], and localized content based image retrieval (LCBIR) [13].

One major assumption of most existing MIL methods is that instances (and bags) are independently and identically distributed. But in many applications, the dependencies between instances/bags naturally exist and if incorporated in models, they can potentially improve the prediction performance significantly. For example, in business analytics, big corporations often analyze the websites of different companies to look for potential partnerships. Since not all of the webpages in a website are useful, we can treat the whole website of a specific company as a bag, and each webpage in this website is considered as an instance. The hyperlinks between webpages provide important information on various relationships between these companies (e.g. supply-demand or joint-selling) and more partner companies can be identified if we follow the hyperlinks of existing partners. Another example is protein fold identification [15], whose goal is to predict protein fold with low conservation in primary sequence, e.g., Thioredoxin-fold (Trx-fold). MIL algorithms have been applied to identify new Trx-fold proteins, where each protein sequence is considered as a bag, and some of its subsequences are instances. The relational information between protein sequences, such as same organism locations or similar species origins, can be used to help the prediction tasks.

Several recent methods have been proposed to model the dependencies between instances in each bag [11, 12, 21]. However, none of them takes into consideration the relational structure between bags or between instances across bags. Furthermore, most of existing MIL research only uses content similarity for modeling structures between instances in each bag, but does not consider other types of relational structure information (e.g. hyperlink) among instances or bags. While much research work [16, 22] for traditional single instance learning has demonstrated that additional structure information (e.g., hyperlink) can be very useful, we believe this is similar for MIL.

Generally speaking, we summarize three scenarios of the structure information in MIL: (1) the relational structures are on the instance level. For example, in the business partner example, the hyperlinks between different webpages can be considered as the relational structure between instances (either in the same bag or across bags). (2) the structure information is available on the bag level. For example, in protein fold identification task, we can consider the phylogenetic tree to capture the evolutionary dependencies between protein sequences . (3) the structure information is available on both instance level and bag level. We refer these three scenarios of learning problems collectively as *multiple instance learning on structured data (MILSD)*.

In this paper, we propose a general framework that address all three structure learning scenarios for MIL. The model consists of a regularization term that confines the capacity of the classifier, a term that penalizes the difference between the predicted labels of the bags and their true labels, and a graph regularization term based on the structure information. The corresponding optimization problem is non-convex. But we show that it can be expressed as the difference between two convex functions. Then, we employ an iterative method – *Constrained Concave-Convex Procedure* (*CCCP*) [14, 19] to solve this problem. To make the proposed method scalable to large datasets, the Cutting Plane method [8] is adapted to solve the subproblems derived from each CCCP iteration. The novelty of the proposed variant of Cutting Plane method lies in modeling dual sets of constraints, *i.e.,* one from modeling instances in individual bags, and the other from the structure information, and its ability to control the precisions (i.e., $\epsilon_1$ and $\epsilon_2$ in Table 1) on different sets of constraints separately. The reason why we need to control precisions separately is that since different sets of constraints normally are derived from various sources and have different forms, their characteristics, as well as the required optimization precisions, are very likely to be diverse. Furthermore, we prove an upper bound of the convergence rate of the proposed optimization method, which is a significant result given our optimization scheme for dual constraint sets can also be applied to many other learning problems. Experiments on three applications demonstrate the advantages of the proposed research.

## 2 Methodology

### 2.1 Problem Statement and Notation

Suppose we are given a set of $n$ labeled bags $\{(\mathbf{B}_i, Y_i), i = 1, 2, \cdots, n\}$, $u$ unlabeled bags $\{\mathbf{B}_i, i = n + 1, n + 2, \cdots, n + u\}$, and a directed or undirected graph $G = (V, E)$ that depicts the structure between either bags or instances. Here, the instances in the bag $\mathbf{B}_i$ are denoted as $\{\mathbf{B}_{i1}, \mathbf{B}_{i2}, ..., \mathbf{B}_{in_i}\} \in \mathcal{X}$, where $n_i$ is the total number of instances in this bag and $Y_i \in \{-1, 1\}$. Each node $v \in V$ corresponds to either a bag or an instance in either the labeled set or the unlabeled set, and the $j$-th edge $e_j = (p, q) \in E$ represents a link from node $p$ to node $q$. The task is to learn a classifier $\mathbf{w}$[1] based on labeled, unlabeled bags, and the predefined structure graph so that the unlabeled bags can be correctly classified. The soft label for the instance $\mathbf{x}$ can be estimated by: $f(\mathbf{x}) = \mathbf{w}^T \mathbf{B}_{ij}$. The soft label of the bag $\mathbf{B}_i$ can be modeled as: $f(\mathbf{B}_i) = \max_{j \in \mathbf{B}_i} \mathbf{w}^T \mathbf{B}_{ij}$, and if $f(\mathbf{B}_i) > 0$, this bag would be labeled as positive and otherwise negative.

### 2.2 Formulation

Our motivation is that labeled bags should be correctly classified and the soft labels of the bags or instances defined on the graph $G$ should be as smooth as possible. Specifically, a pair of nodes linked by an edge tend to possess the same label and therefore the nodes lying on a densely linked subgraph are likely to have the same labels [20]. The general formulation of MILSD is given as:

$\min_{\mathbf{w}} H_r(\mathbf{w}) + H_d(\mathbf{w}) + H_G(\mathbf{w})$, where $H_r(\mathbf{w})$ is a regularization term based on $\mathbf{w}$, and depicts the capacity of this classifier. One of the possible options, which is also the one used in this paper, is $\|\mathbf{w}\|^2$. $H_d(\mathbf{w})$ penalizes the difference between the estimated bag labels and the given labels. In this paper, without loss of generality, the hinge loss is used [3]. So, given a classifier $\mathbf{w}$, $H_d(\mathbf{w})$ is calculated as: $\frac{C}{n} \sum_{i=1}^{n} \max\{0, 1 - \max_{j \in \mathbf{B}_i} Y_i \mathbf{w}^T \mathbf{B}_{ij}\}$, where $C$ is the trade off parameter. $H_G(\mathbf{w})$ is a graph regularization term based on the given graph $G$ that enforces the smoothness on the soft labels of nodes in the given graph, which can be defined as: $\frac{\mu}{|E|} \sum_{(p,q) \in E} w(p,q) \left| \frac{f(v_p)}{\sqrt{d(p)}} - \frac{f(v_q)}{\sqrt{d(q)}} \right|$, where $v_p$ and $v_q$ are two nodes in the graph. $w(p,q)$ is a weight function that measures the weight on the edge $(p,q)$. $d(p)$ and $d(q)$ are the outgoing degrees for the node $v_p$ and $v_q$ respectively [20]. $|E|$ is the number of edges in graph $G$. Depending on which one of the three scenarios the graph is defined, we name the formulation where the graph is defined on instances as I-MILSD, the formulation where the graph is defined on bags as B-MILSD, and the formulation where the graph is defined on both bags and instances as BI-MILSD. In particular,

1. For I-MILSD, $H_G(\mathbf{w})$ can be defined as $\frac{\mu}{|E|} \sum_{(p,q) \in E} w(p,q) \left| \frac{\mathbf{w}^T \mathbf{x}_p}{\sqrt{d(p)}} - \frac{\mathbf{w}^T \mathbf{x}_q}{\sqrt{d(q)}} \right|$, where $\mathbf{x}_p$ and $\mathbf{x}_q$ are two instances.

2. For B-MILSD, $H_G(\mathbf{w})$ is defined as $\frac{\mu}{|E|} \times \sum_{(p,q) \in E} w(p,q) \left| \frac{\max_{j \in \mathbf{B}_p} \mathbf{w}^T \mathbf{B}_{pj}}{\sqrt{d(p)}} - \frac{\max_{j \in \mathbf{B}_q} \mathbf{w}^T \mathbf{B}_{qj}}{\sqrt{d(q)}} \right|$ is defined. Then, the B-MILSD problem can be formulated as follows:

$$\min_{\mathbf{w}} \quad \frac{1}{2}\|\mathbf{w}\|^2 + \frac{C}{n}\sum_{i=1}^{n}\xi_i + \frac{\mu}{|E|} \times \sum_{(p,q) \in E} w(p,q) \left| \frac{\max_{j \in B_p} \mathbf{w}^T \mathbf{B}_{pj}}{\sqrt{d(p)}} - \frac{\max_{j \in B_q} \mathbf{w}^T \mathbf{B}_{qj}}{\sqrt{d(q)}} \right|$$
$$s.t. \quad \forall i \in \{1, 2, \ldots, n\}, \quad Y_i \max_{j \in B_i} \mathbf{w}^T \mathbf{B}_{ij} \geq 1 - \xi_i, \tag{1}$$

where $\xi_i$ is the hinge loss and $\mu$ is the trade-off parameter for the graph term. The formulation proposed in [1] is a special case of the proposed formulation, with $\mu$ equals zero.

3. The definition of $H_G(\mathbf{w})$ in BI-MILSD can be considered as a combination of previous two formulations.

In the following sections, our focus will be on the more challenging problem as B-MILSD, while I-MILSD and BI-MILSD can be solved in a similar way, since the $H_G(\mathbf{w})$ in I-MILSD is convex and the formulation of BI-MILSD can be considered as a combination of the B-MILSD and I-MILSD.

## 2.3 Optimization Procedure with CCCP and Multi-Constraint Cutting Plane

The formulation in problem (1) combines both the goodness-of-fit for labeled bags and the structure information embedded in the graph. However, since both $H_G(\mathbf{w})$ and the constraints in problem (1) are non-convex, the global optimal solution of this problem cannot be attained. To solve this problem, the constrained concave-convex procedure (CCCP) is used. It is an optimization method that deals with the concave convex objective function with concave convex constraints [14]. In this paper, without loss of generality, we only assume $w(p,q)$ to be a canonical weight function. To employ CCCP, first of all, for each edge $(p,q)$, a non-negative loss variable $\zeta_{(p,q)}$ is introduced. Then, problem (1) can be solved iteratively. In particular, given an initial point $\mathbf{w}^{(0)}$, CCCP iteratively computes $\mathbf{w}^{(t+1)}$ from $\mathbf{w}^{(t)}$ [2] by replacing $\max_{j \in \mathbf{B}_i} \mathbf{w}^T \mathbf{B}_{ij}$ with its first order Taylor expansions at $\mathbf{w}^{(t)}$, and solving the resulting quadratic programming problem as follows, until convergence $(u_i^{(t)} = \arg\max_{j \in \{1,\ldots,n_i\}} (\mathbf{w}^{(t)})^T \mathbf{B}_{ij})$.

$$\min_{\mathbf{w},\xi_i\geq0,\zeta_{(p,q)}\geq0} \quad \frac{1}{2}\|\mathbf{w}\|^2 + \frac{C}{n}\sum_{i=1}^{n}\xi_i + \frac{\mu}{|E|}\sum_{(p,q)\in E}\zeta_{(p,q)} \tag{2}$$

$$s.t. \quad \forall i \in \{1,2,\ldots,n\}, \quad Y_i\mathbf{w}^T\mathbf{B}_{iu_i^{(t)}} \geq 1-\xi_i$$

$$\forall(p,q)\in E, \quad \forall k\in\{1,2,\ldots,n_p\}, \frac{\mathbf{w}^T\mathbf{B}_{pk}}{\sqrt{d(p)}} - \frac{\mathbf{w}^T\mathbf{B}_{qu_q^{(t)}}}{\sqrt{d(q)}} \leq \zeta_{(p,q)}$$

$$\forall k\in\{n_p+1,\ldots,n_p+n_q\}, \frac{\mathbf{w}^T\mathbf{B}_{q(k-n_p)}}{\sqrt{d(q)}} - \frac{\mathbf{w}^T\mathbf{B}_{pu_p^{(t)}}}{\sqrt{d(p)}} \leq \zeta_{(p,q)}.$$

The problem (2) can be directly solved as a standard quadratic programming problem [2]. However, in many real world applications, the number of the labeled bags as well as the number of links between bags are huge. In this case, we would need to find a way that can solve this problem efficiently. Instead of directly solving this optimization problem, we employ the Cutting Plane method [8], which has shown its effectiveness and efficiency in solving similar tasks recently [6]. But different from the method employed in [6], in this paper, we need to deal with two sets of constraints, rather than just one constraint set, with specified precisions separately. A new way to adapt the Cutting Plane method is devised here. Problem (2) is equivalently transformed to the following form:

$$\min_{\mathbf{w},\xi\geq0,\zeta\geq0} \quad \frac{1}{2}\|\mathbf{w}\|^2 + C\xi + \mu\zeta \tag{3}$$

$$s.t. \ \forall\mathbf{c}\in\{0,1\}^n, \frac{1}{n}\mathbf{w}^T\sum_{i=1}^{n}\mathbf{c}_iY_i\mathbf{B}_{iu_i^{(t)}} \geq \frac{1}{n}\sum_{i=1}^{n}\mathbf{c}_i - \xi$$

$$\forall(\tau\in\{0,1\}^{|E|\times(n_p+n_q)})\bigcap(\forall e_j\in E, \sum_{k=1}^{n_p+n_q}\tau_{jk}\leq1)$$

$$\frac{\mathbf{w}^T}{|E|}\sum_{j=1}^{|E|}\left(\sum_{k=1}^{n_p}\tau_{jk}(\frac{\mathbf{B}_{pk}}{\sqrt{d(p)}} - \frac{\mathbf{B}_{qu_q^{(t)}}}{\sqrt{d(q)}}) + \sum_{k=1}^{n_q}\tau_{j(k+n_p)}(\frac{\mathbf{B}_{qk}}{\sqrt{d(q)}} - \frac{\mathbf{B}_{pu_p^{(t)}}}{\sqrt{d(p)}})\right) \leq \zeta,$$

where, $e_j = (p,q)$, $\tau$ is a matrix with $|E|$ rows and a varying number of columns: for the $j$-th row of $\tau$, it has $n_p + n_q$ columns (possible constraints). For each edge, at most one constraint could be activated for each feasible $\tau$.

**Theorem 1:** Any solution $\mathbf{w}^*$ of problem (2) is also a solution to problem (3) (and vice versa) with $\xi^* = \frac{1}{n}\sum_{i=1}^{n}\xi_i^*$ and $\zeta^* = \frac{1}{|E|}\sum_{(p,q)\in E}\zeta_{(p,q)}^*$ [3].
**Proof:** Please refer to the supplemental materials in the author's homepage.

The benefit of making this transformation is that, as we shall see later, during each Cutting Plane iteration at most two constraints will be added and therefore the final solution would be extremely sparse, with the number of non-zero dual variables independent of the number of training examples. Now the problem turns to how to solve the problem (3) efficiently, which is convex, but contains two sets of exponential number of constraints due to the large number of feasible $\mathbf{c}$ and $\tau$. We present a novel adaption of the Cutting Plane method that can handle the two sets of constraints simultaneously. More specifically, the main motivation of the method proposed here is to find two small subsets of constraints, *i.e.,* $\Omega_1$ and $\Omega_2$ from constraint sets in Eq.(3). With these two sets of selected constraints, the solution of the corresponding relaxed problem satisfies all the constraints from problem (3) up to two precisions $\epsilon_1$ and $\epsilon_2$, *i.e.,* $\forall\mathbf{c}\in\{0,1\}^n$: $\frac{1}{n}\mathbf{w}^T\sum_{i=1}^{n}\mathbf{c}_iY_i\mathbf{B}_{iu_i^{(t)}} \geq \frac{1}{n}\sum_{i=1}^{n}\mathbf{c}_i - (\xi+\varepsilon_1)$ and $\forall(\tau\in\{0,1\}^{|E|\times(n_p+n_q)})\bigcap(\forall e_j\in E, \sum_{k=1}^{n_p+n_q}\tau_{jk}\leq 1)$: $\frac{1}{|E|}\mathbf{w}^T\sum_{j=1}^{|E|}\left(\sum_{k=1}^{n_p}\tau_{jk}(\frac{\mathbf{B}_{pk}}{\sqrt{d(p)}} - \frac{\mathbf{B}_{qu_q^{(t)}}}{\sqrt{d(q)}}) + \sum_{k=1}^{n_q}\tau_{j(k+n_p)}(\frac{\mathbf{B}_{qk}}{\sqrt{d(q)}} - \frac{\mathbf{B}_{pu_p^{(t)}}}{\sqrt{d(p)}})\right) \leq (\zeta+\varepsilon_2)$. It indicates that the two remaining sets of constraints (that are not added to $\Omega_1$ and $\Omega_2$) will not be violated up to two precisions $\varepsilon_1$ and $\varepsilon_2$ respectively, and therefore they do not need to be added to $\Omega_1$ and $\Omega_2$ explicitly.

The proposed method constructs $\Omega_1^t$ and $\Omega_2^t$ iteratively, which starts from two empty sets $\Omega_1^{t_0}$ [4] and $\Omega_2^{t_0}$ respectively. During the $s$-th Cutting Plane iteration, based on the $\mathbf{w}^{t_s}$, the most violated constraint for $\Omega_1^{t_s}$ can be computed as:

$$
\mathbf{c}_i^{t_s} = \begin{cases} 1, & \text{if } Y_i(\mathbf{w}^{t_s})^T \mathbf{B}_{iu_i^{(t)}} < 1 \\ 0, & otherwise \end{cases},
\tag{4}
$$

and the most violated constraint for $\Omega_2^{t_s}$ can be computed as:

$$
\tau_{jk}^{t_s} = \begin{cases} 1, & \text{if}(k = k^*) \bigcap (\max\left\{ \max_{k\in\{1,\dots,n_p\}}(\mathbf{w}^{t_s})^T(\frac{\mathbf{B}_{pk}}{\sqrt{d(p)}} - \frac{\mathbf{B}_{qu_q^{(t)}}}{\sqrt{d(q)}}), \max_{k\in\{n_p+1,\dots,n_p+n_q\}}(\mathbf{w}^{t_s})^T(\frac{\mathbf{B}_{q(k-n_p)}}{\sqrt{d(q)}} - \frac{\mathbf{B}_{pu_p^{(t)}}}{\sqrt{d(p)}}) \right\} > 0) \\ 0, & otherwise, \end{cases}
\tag{5}
$$

$$
k^* = \arg\max_k \left\{ \max_{k\in\{1,\dots,n_p\}}(\mathbf{w}^{t_s})^T(\frac{\mathbf{B}_{pk}}{\sqrt{d(p)}} - \frac{\mathbf{B}_{qu_q^{(t)}}}{\sqrt{d(q)}}), \max_{k\in\{n_p+1,\dots,n_p+n_q\}}(\mathbf{w}^{t_s})^T(\frac{\mathbf{B}_{q(k-n_p)}}{\sqrt{d(q)}} - \frac{\mathbf{B}_{pu_p^{(t)}}}{\sqrt{d(p)}}) \right\}.
$$

After calculating these two sets of most violated constraints, the two stopping conditions can be computed:

$$
H_1^{t_s} = \left( \frac{(\mathbf{w}^{t_s})^T}{n} \sum_{i=1}^n \mathbf{c}_i^{t_s} Y_i \mathbf{B}_{iu_i^{(t)}} \geq \frac{1}{n}\sum_{i=1}^n \mathbf{c}_i^{t_s} - (\xi^{t_s} + \varepsilon_1) \right),
\tag{6}
$$

$$
H_2^{t_s} = \left( \frac{(\mathbf{w}^{t_s})^T}{|E|} \times \sum_{j=1}^{|E|} \left( \sum_{k=1}^{n_p}\tau_{jk}^{t_s}(\frac{\mathbf{B}_{pk}}{\sqrt{d(p)}} - \frac{\mathbf{B}_{qu_q^{(t)}}}{\sqrt{d(q)}}) + \sum_{k=n_p+1}^{n_p+n_q}\tau_{j(k-n_p)}^{t_s}(\frac{\mathbf{B}_{qk}}{\sqrt{d(q)}} - \frac{\mathbf{B}_{pu_p^{(t)}}}{\sqrt{d(p)}}) \right) \leq (\zeta^{t_s} + \varepsilon_2)).
\tag{7}
$$

The Cutting Plane iteration will terminate if both conditions $H_1^{t_s}$ and $H_2^{t_s}$ are true. Otherwise, $\mathbf{c}^{t_s}$ will be added to $\Omega_1^{t_s}$ if $H_1^{t_s}$ is false, and $\tau^{t_s}$ will be added to $\Omega_2^{t_s}$ if $H_2^{t_s}$ is false. Then, the new optimization problem turns to:

$$
\min_{\mathbf{w},\xi\geq 0,\zeta\geq 0} \quad \frac{1}{2}\|\mathbf{w}\|^2 + C\xi + \mu\zeta
\tag{8}
$$

$$
s.t. \quad \forall \mathbf{c} \in \Omega_1^{t_s}, \quad \frac{1}{n}\mathbf{w}^T\sum_{i=1}^n \mathbf{c}_i Y_i \mathbf{B}_{iu_i^{(t)}} \geq \frac{1}{n}\sum_{i=1}^n \mathbf{c}_i - \xi
$$

$$
\forall \tau \in \Omega_2^{t_s}, \quad \frac{\mathbf{w}^T}{|E|}\sum_{j=1}^{|E|}\left( \sum_{k=1}^{n_p}\tau_{jk}(\frac{\mathbf{B}_{pk}}{\sqrt{d(p)}} - \frac{\mathbf{B}_{qu_q^{(t)}}}{\sqrt{d(q)}}) + \sum_{k=1}^{n_q}\tau_{j(k+n_p)}(\frac{\mathbf{B}_{qk}}{\sqrt{d(q)}} - \frac{\mathbf{B}_{pu_p^{(t)}}}{\sqrt{d(p)}}) \right) \leq \zeta.
$$

This optimization problem can be solved efficiently through the dual form [2].

## 2.4 Analysis and Discussions

The whole algorithm of B-MILSD is described in Table 1. Here, $J^t = \frac{1}{2}\|\mathbf{w}^{(t)}\|^2 + C\xi^{(t)} + \mu\zeta^{(t)}$. The convergence of the proposed method is guaranteed. Given an initial $\mathbf{w}$, the outer CCCP iteration has already been proved to converge to a local optimal solution [14]. The final solution can be improved by running this algorithm several times and picking the solution with the smallest $J^{(t)}$ value. We will show that the Cutting Plane iterations with two different sets of constraints converge in a fixed number of steps through the following two theorems.

**Theorem 2**: For each Cutting Plane iteration described in Table 1, the objective function of (8) will be increased by at least $\kappa = \min\{\frac{C\epsilon_1}{2}, \frac{\epsilon_1^2}{8R^2}, \frac{\mu\epsilon_2}{2}, \frac{\epsilon_2^2}{16R^2}, \frac{(\epsilon_1+\epsilon_2)^2}{(24+16\sqrt{2})R^2}\}$, where $R^2 = \max_{i,j}\mathbf{B}_{ij}^2$.
**Sketch of Proof:** The detailed proof of this theorem can be found in the supplemental materials. Here, we only briefly outline the way how we proved it. In each Cutting Plane iteration described in Table 1, there are three possibilities for updating the constraints. In each case, we will find a feasible direction for increasing the objective function. A line search method will then be used to

Table 1: The description of B-MILSD

**CCCP Iterations**:
1. Initialize $\mathbf{w}^0$, t=0, $\Delta J = 10^3$, $J^{-1} = 10^3$.
2. **while** $\Delta J / J^{t-1} > \delta$ **do**
3.   Derive problem (2). Set the constraint set $\Omega_1^{t0} = \phi$, $\Omega_2^{t0} = \phi$ and $s = -1$.
   **Cutting Plane Iterations:**
4.   **repeat**
5.     $s = s+1$.
6.     Get $(\mathbf{w}^{(ts)}, \xi^{(ts)}, \zeta^{(ts)})$ by solving (8).
7.     Compute the most constraints, i.e., $\mathbf{c}^{ts}$, and $\tau^{ts}$ by Eq.(4) and Eq.(5).
8.     Compute the stopping criteria, i.e., $H_1^{ts}$, and $H_2^{ts}$ by Eq.(6) and Eq.(7).
9.     Update $\Omega_1^{ts}$ by $\Omega_1^{t(s+1)} = \Omega_1^{ts} \bigcup \mathbf{c}^{ts}$, if $H_1^{ts}$ is false. Otherwise, $\Omega_1^{t(s+1)} = \Omega_1^{ts}$. Update $\Omega_2^{ts}$ by $\Omega_2^{t_s+1} = \Omega_2^{ts} \bigcup \tau^{ts}$ if $H_2^{ts}$ is false. Otherwise, $\Omega_2^{t(s+1)} = \Omega_2^{ts}$.
10.    **while** $H_1^{ts} \bigwedge H_2^{ts}$ is false
11.  $t = t+1$.
12.  $\mathbf{w}^{(t)} = \mathbf{w}^{(t-1)s}$, $\xi^{(t)} = \xi^{(t-1)s}$, and $\zeta^{(t)} = \zeta^{((t-1)s}$.
13.  $\Delta J = J^{t-1} - J^t$.
14. **end while**

find the optimal increment, which serves as the lower bound for each updating. (1) $H_1^{ts}$ is false. $H_2^{ts}$ is true. $\mathbf{c}^{ts}$ is added to $\Omega_1^{ts}$. The minimal improvement of the objective function for problem (8) after this constraint is added would be $\min\{\frac{C\epsilon_1}{2}, \frac{\epsilon_1^2}{8R^2}\}$. (2) $H_1^{ts}$ is true. $H_2^{ts}$ is false. $\Omega_2^{ts}$ is updated by appending $\tau^{ts}$. In this case, the minimal increment will be $\min\{\frac{\mu\epsilon_2}{2}, \frac{\epsilon_2^2}{16R^2}\}$. (3) Both $H_1^{ts}$ and $H_2^{ts}$ are false. The most violated constraints are added to both $\Omega_1^{ts}$ and $\Omega_2^{ts}$. We proved that the minimal increment is $\min\{\frac{\min\{C,\mu\}(\epsilon_1+\epsilon_2)}{2}, \frac{(\epsilon_1+\epsilon_2)^2}{(24+16\sqrt{2})R^2}\}$. By integrating all of these three cases, it is clear that for each Cutting Plane iteration, the minimal increment is $\kappa = \min\{\frac{C\epsilon_1}{2}, \frac{\epsilon_1^2}{8R^2}, \frac{\mu\epsilon_2}{2}, \frac{\epsilon_2^2}{16R^2}, \frac{(\epsilon_1+\epsilon_2)^2}{(24+16\sqrt{2})R^2}\}$, since $\min\{\frac{C\epsilon_1}{2}, \frac{\mu\epsilon_2}{2}\} \leq \frac{\min\{C,\mu\}(\epsilon_1+\epsilon_2)}{2}$.

**Theorem 3**: The proposed Cutting Plane iteration terminates after at most $\frac{C}{\kappa}$ steps, where, $\kappa = \min\{\frac{C\epsilon_1}{2}, \frac{\epsilon_1^2}{8R^2}, \frac{\mu\epsilon_2}{2}, \frac{\epsilon_2^2}{16R^2}, \frac{(\epsilon_1+\epsilon_2)^2}{(24+16\sqrt{2})R^2}\}$, and $R^2 = \max_{i,j} \mathbf{B}_{ij}^2$.

**Proof:** $\mathbf{w} = 0, \xi = 1, \zeta = 0$ is a feasible solution for problem (3). Therefore, the objective function of (3) is upper bounded by $C$, and should be lower bounded by $0$. Given the conclusion from Theorem 2, it is clear that the Cutting Plane iteration will terminate within $\frac{C}{\kappa}$ steps.

The Cutting Plane method has already been employed in several previous works. In [6, 7, 17], the authors adapted the Cutting Plane method to accelerate structural SVM related algorithms. However, these works do not explicitly consider the case when several different sets of constraints with specified precisions are involved. The novelty of the proposed method lies on its ability to control these optimization precisions separately, and meanwhile it still enjoys the sparseness of the final solution with respect to the number of dual variables, which is brought by slack variable transformation.

In [18], the authors solved the problem of structural SVM with latent variables by employing CCCP and the bundle method [9]. MIL problem itself can be considered as a special case of latent variable problem. But the major limitation of [18] is that they cannot incorporate the relational information into the formulation, and therefore cannot be used here. Furthermore, [18] does not consider dual sets of constraints in optimization, which is less appropriate than the proposed optimization method.

## 3 Experiments

### 3.1 Webpage Classification

In webpage classification, each webpage can be considered as a bag, and its corresponding passages represent its instances [1]. The hyperlinks between different webpages are treated as the additional relational structure/links between different bags. WebKB[5] dataset is used in experiments. There are

Figure 1: Classification and CPU Time Comparisons

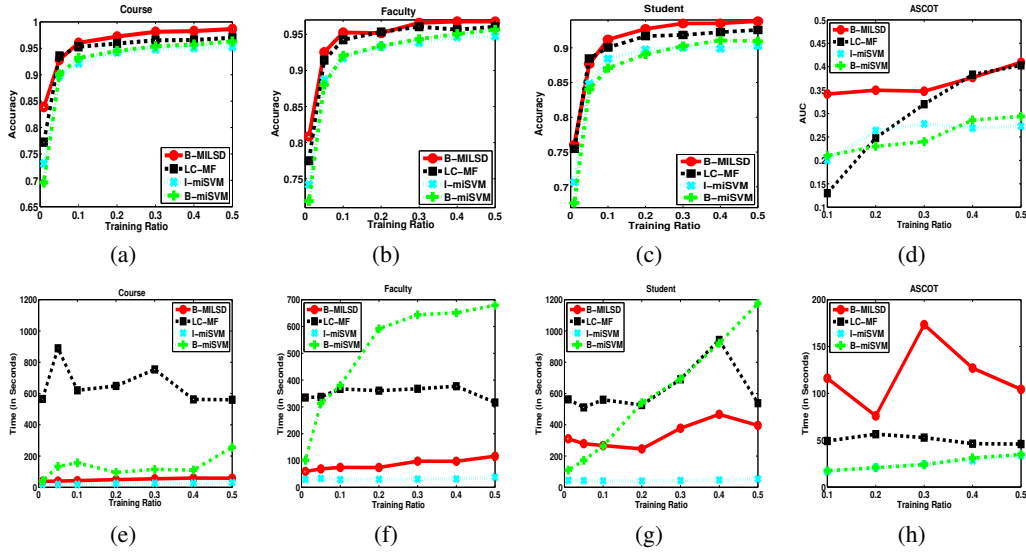

in total 8280 webpages in this dataset. The webpages without any incoming and outgoing links are deleted, and 6883 webpages are left. The three most frequently appeared categories, i.e., student, course, and faculty, are used for classification, where each sub-dataset contains all of the webpages/bags from one of the three categories, and the same number of the negative bags randomly sampled from the remaining six categories in WebKB. The hyperlinks between these webpages are used as the structure/link information. The tf-idf (normalized term frequency and log inverse document frequency) [10] features are extracted for each passage, and the stop words are removed. We use porter as the stemmer.

In the proposed method, $C$ and $\mu$ are set by 5-fold cross validation through the grids $2^{[-5:5]}$ and $[0, 0.01, 0.1, 1]$ respectively on the training set. To show the effects of the structure information on the performance of MIL methods, we compare the proposed method with the instance-based multiple instance support vector machine (I-miSVM) as well as the bag-based multiple instance support vector machine (B-miSVM) [1]. The formulation of these two methods can be considered as a special case of the proposed method with $\mu$ equals zero, and they are two different heuristic iterative ways of implementing the same formulation. Their parameters are also set by 5 fold cross validations. Link-Content Matrix Factorization (LC-MF) is a non-MIL matrix factorization method [22], which has been shown to outperform several alternatives, including SVM. We conduct experiments with LC-MF based on the single instances that we extract for the same set of examples, and the corresponding links. Similar to [22], the number of latent factors is set to be 50. After computing the latent factors, a linear SVM is trained on the training set with the hinge loss parameter $C$ being determined by using 5-fold cross validation. For each experiment, a fixed ratio of bags are chosen as the training set, while the remaining examples are treated as the testing set. The average results over 20 independent runs are reported on the training ratios $[0.01, 0.05, 0.1, 0.2, 0.3, 0.4, 0.5]$.

The classification results are reported in Fig.1(a)(b)(c) and the CPU time comparison results are announced in Fig.1(e)(f)(g). In Table 2, we further report the performances when the training ratio equals 0.2. From these experimental results, it is clear that the performance of the proposed method is better than the other comparison methods in accuracy and its CPU time is comparatively low.

## 3.2 Market Targeting

Market targeting is a popular topic for big corporations. Its basic objective is to automatically identify potential partners. One of the feasible market targeting strategy is to analyze the websites of the potential partners. But usually not all of the webpages are useful for partner identification. So, it is better to formulate it as a MIL problem, in which each website is considered as a bag, and its

associated webpages are considered as instances. Two related companies may be connected through hyperlinks in some of their webpages.

We obtained a dataset (ASCOT) from a big international corporation. In ASCOT, the webpages of around 225 companies are crawled. 25 of the companies/bags are labeled as positive, since they are partners of this corporation, while the remaining 200 companies/bags are treated as negative ones. For each company, the webpages with less than 100 unique words are removed and at most 50 webpages with the largest number of unique words[6] are selected as instances. The hyperlinks between webpages of different companies are treated as the structure information. For each experiment, we fix the training ratio of positive and negative bags, while the remaining bags are considered as the testing set. The averaged results over 20 independent runs are reported on the training ratios $[0.1, 0.2, 0.3, 0.4, 0.5]$. The parameters for different methods are tuned in the same way as on WebKB. But for the ratios $0.1$ and $0.2$, we use 3-fold cross validation due to the lack of positive bags. For LC-MF, experiments are conducted on the instances which are the averages of the instances in each bag. Because of the extremely imbalanced nature of this dataset, the Area Under Curve (AUC) is used as the measure criteria.

The corresponding results are reported in Fig.1(d) and Fig.1(h). In Table 2, we report the performances when the training ratio equals $0.2$. On this dataset, B-MILSD performs much better than the comparison methods, especially when the ratio of training examples is low. This is because the hyperlink information helps a lot when the content information is rare in MIL, and the MIL setting is useful to eliminate the useless instances especially when the supervised information is scare.

### 3.3 Protein Fold Identification

In protein fold identification [15], the low conservation of primary sequence in protein superfamilies such as Thioredoxin-fold (Trx-fold) makes conventional modeling methods, such as Hidden Markov Models difficult to use. MIL can be used to identify new Trx-fold proteins naturally, in which each protein sequence is considered as a bag, and some of its subsequences are considered as instances. Here, we use a benchmark protein dataset[7]. In each protein's primary sequence, first of all, the primary sequence motif (typically CxxC) are found. Then, a window of size 214 around it are extracted and aligned. These windows are then mapped to a 8-dimensional feature space. The similarities between different proteins are estimated by using clustalw[8]. If the score between a pair of proteins exceed 25, then we consider there exists a link between them.

Following the experiment setting in [15], we conduct 5 fold cross validation to test the performances. The averaged classification accuracies and CPU Running Time are reporeted in Table 2. From the comparison methods, we can see that on this dataset, the proposed method is both efficient and effective. Its CPU running time is almost $10 - 100$ times faster than the comparison methods.

Table 2: Performance Comparisons

|  | Measure | B-MILSD | LC-MF | I-miSVM | B-miSVM |  | Measure | B-MILSD | LC-MF | I-miSVM | B-miSVM |
|---|---|---|---|---|---|---|---|---|---|---|---|
| Course | Accuracy (%) | **97.2** | 95.9 | 94.3 | 94.5 | ASCOT | AUC | **0.350** | 0.248 | 0.264 | 0.230 |
|  | Time (seconds) | 49.1 | 648.5 | 23.9 | 95.6 |  | Time (seconds) | 76.0 | 56.4 | 20.9 | 20.7 |
| Faculty | Accuracy (%) | **95.2** | 95.3 | 93.3 | 93.4 | Protein | Accuracy (%) | **96.2** | 95.2 | 92.2 | 82.7 |
|  | Time (seconds) | 73.9 | 360.6 | 29.7 | 591.6 |  | Time (seconds) | 1.7 | 16.9 | 160.3 | 73.8 |
| Student | Accuracy (%) | **92.7** | 91.7 | 89.5 | 89.1 |  |  |  |  |  |  |
|  | Time (seconds) | 245.7 | 526.3 | 41.2 | 540.4 |  |  |  |  |  |  |

## 4 Conclusions

This paper presents a novel machine learning problem – multiple instance learning on structured data (MILSD) for incorporating additional structure information into multiple instance learning. In particular, a general framework of MILSD is proposed for dealing with the additional structure information in different scenarios. An effective and efficient optimization method is proposed for MILSD by combining the CCCP method and a new multi-constraint Cutting Plane method. Some theoretical results are proved to justify the methodology that we employed to handle multi-sets of

constraints with the Cutting Plane method. The experimental results on three different applications clearly demonstrate the advantages of the proposed method. For future work, we plan to adapt the current framework to solve multi-view multiple instance learning on structured data.

**Acknowledgement:** The work of Dan Zhang and Luo Si was partially supported by NSF research grant IIS-0746830, CNS-1012208, IIS-1017837, and the Center for Science of Information (CSoI), an NSF Science and Technology Center, under grant agreement CCF-0939370. The work of Yan Liu was partially sponsored by the U.S. Defense Advanced Research Projects Agency (DARPA) under the Anomaly Detection at Multiple Scales (ADAMS) program, Agreement Number W911NF-11-C-0200. The authors would also like to express their sincere thanks to Prof. S.V.N. Vishwanathan and the anonymous reviewers for their constructive suggestions.

## Footnotes

[1]Without loss of generality, in this paper, we only consider linear classifiers. Here, the bias of the classifier is absorbed by the feature vectors. The kernel version [3] of the proposed method can be easily derived.

[2]The superscript $t$ is used to denote that the result is obtained from the $t$-th CCCP iteration. For example, $\mathbf{w}^{(t)}$ is the optimized classifier from the $t$-th CCCP iteration step.

[3]the subscript $*$ denotes the optimal value of the corresponding variable.

[4] Here, $t_s$ denotes the $s$-th Cutting Plane iteration for solving the problem from the $t$-th CCCP iteration.

[5]http://www.cs.cmu.edu/~webkb/

[6]Still, we use porter as the stemmer and have removed the stop words.

[7]http://cse.unl.edu/∼ qtao/datasets/mil_dataset_Trx_protein.html

[8]http://www.ebi.ac.uk/Tools/msa/clustalw2/

## References

[1] S. Andrews, I. Tsochantaridis, and T. Hofmann. Support vector machines for multiple-instance learning. In *NIPS*, 2003.

[2] S.P. Boyd and L. Vandenberghe. *Convex optimization*. Cambridge Univ Press, 2004.

[3] B.Scholkopf and A.Smola. *Learning with Kernels*. MITPress, Cambridge, MA, 2002.

[4] T. G. Dietterich, R. H. Lathrop, and T. Lozano-Perez. Solving the multiple instance problem with axis-parallel rectangles. In *Artificial Intelligence*, 1998.

[5] T. Gärtner, P.A. Flach, A. Kowalczyk, and A.J. Smola. Multi–instance kernels. In *ICML*, 2002.

[6] T. Joachims. Training linear SVMs in linear time. In *KDD*, 2006.

[7] T. Joachims, T. Finley, and C.N.J. Yu. Cutting-plane training of structural SVMs. *Machine Learning*, 2009.

[8] JE Kelley Jr. The cutting-plane method for solving convex programs. *JSIAM*, 1960.

[9] Krzysztof C. Kiwiel. Proximity control in bundle methods for convex nondifferentiable minimization. *Math. Program.*, 46:105–122, 1990.

[10] Christopher D. Manning, Prabhakar Raghavan, and Hinrich Schtze. *Introduction to Information Retrieval*. Cambridge University Press, 2008.

[11] Amy McGovern and David Jensen. Identifying predictive structures in relational data using multiple instance learning. In *ICML*, 2003.

[12] G.J. Qi, X.S. Hua, Y. Rui, T. Mei, J. Tang, and H.J. Zhang. Concurrent multiple instance learning for image categorization. In *CVPR*, 2007.

[13] R. Rahmani and S.A. Goldman. MISSL: Multiple-instance semi-supervised learning. In *ICML*, 2006.

[14] A.J. Smola, SVN Vishwanathan, and T. Hofmann. Kernel methods for missing variables. In *AISTATS*, 2005.

[15] Qingping Tao, Stephen D. Scott, N. V. Vinodchandran, and Thomas Takeo Osugi. Svm-based generalized multiple-instance learning via approximate box counting. In *ICML*, 2004.

[16] Benjamin Taskar, Carlos Guestrin, and Daphne Koller. Max-margin markov networks. In *NIPS*, 2003.

[17] I. Tsochantaridis, T. Joachims, T. Hofmann, and Y. Altun. Large margin methods for structured and interdependent output variables. *JMLR*, 2006.

[18] Chun-Nam John Yu and T. Joachims. Learning structural svms with latent variables. In *ICML*, 2009.

[19] A. Yuille and A. Rangarajan. The concave-convex procedure. *Neural Computation*, 2003.

[20] D. Zhou, J. Huang, and B. Scholkopf. Learning from labeled and unlabeled data on a directed graph. In *ICML*, 2005.

[21] Z-H Zhou, Y-Y Sun, and Y-F Li. Multi-instance learning by treating instances as non i.i.d. samples. In *ICML*, 2009.

[22] S-H Zhu, K. Yu, Y. Chi, and Y-H Gong. Combining content and link classification using matrix factorization. In *SIGIR*, 2007.

